# Solitaire: Man Versus Machine

**Xiang Yan**[*]    **Persi Diaconis**[*]    **Paat Rusmevichientong**[†]    **Benjamin Van Roy**[*]

[*]**Stanford University**
{xyan,persi.diaconis,bvr}@stanford.edu

[†]**Cornell University**
paatrus@orie.cornell.edu

## Abstract

In this paper, we use the rollout method for policy improvement to analyze a version of Klondike solitaire. This version, sometimes called *thoughtful solitaire*, has all cards revealed to the player, but then follows the usual Klondike rules. A strategy that we establish, using iterated rollouts, wins about twice as many games on average as an expert human player does.

## 1 Introduction

Though proposed more than fifty years ago [1, 7], the effectiveness of the policy improvement algorithm remains a mystery. For discounted or average reward Markov decision problems with $n$ states and two possible actions per state, the tightest known worst-case upper bound in terms of $n$ on the number of iterations taken to find an optimal policy is $O(2^n/n)$ [9]. This is also the tightest known upper bound for deterministic Markov decision problems. It is surprising, however, that there are no known examples of Markov decision problems with two possible actions per state for which more than $n + 2$ iterations are required. A more intriguing fact is that even for problems with a large number of states – say, in the millions – an optimal policy is often delivered after only half a dozen or so iterations.

In problems where $n$ is enormous – say, a googol – this may appear to be a moot point because each iteration requires $\Omega(n)$ compute time. In particular, a policy is represented by a table with one action per state and each iteration improves the policy by updating each entry of this table. In such large problems, one might resort to a suboptimal heuristic policy, taking the form of an algorithm that accepts a state as input and generates an action as output. An interesting recent development in dynamic programming is the rollout method. Pioneered by Tesauro and Galperin [13, 2], the rollout method leverages the policy improvement concept to amplify the performance of any given heuristic. Unlike the conventional policy improvement algorithm, which computes an optimal policy off-line so that it may later be used in decision-making, the rollout method performs its computations on-line at the time when a decision is to be made. When making a decision, rather than applying the heuristic policy directly, the rollout method computes an action that would result from an iteration of policy improvement applied to the heuristic policy. This does

not require $\Omega(n)$ compute time since only one entry of the table is computed.

The way in which actions are generated by the rollout method may be considered an alternative heuristic that improves on the original. One might consider applying the rollout method to this new heuristic. Another heuristic would result, again with improved performance. Iterated a sufficient number of times, this process would lead to an optimal policy. However, iterating is usually not an option. Computational requirements grow exponentially in the number of iterations, and the first iteration, which improves on the original heuristic, is already computationally intensive. For this reason, prior applications of the rollout method have involved only one iteration [3, 4, 5, 6, 8, 11, 12, 13]. For example, in the interesting study of Backgammon by Tesauro and Galperin [13], moves were generated in five to ten seconds by the rollout method running on configurations of sixteen to thirty-two nodes in a network of IBM SP1 and SP2 parallel-RISC supercomputers with parallel speedup efficiencies of 90%. A second iteration of the rollout method would have been infeasible – requiring about six orders of magnitude more time per move.

In this paper, we apply the rollout method to a version of solitaire, modeled as a deterministic Markov decision problem with over 52! states. Determinism drastically reduces computational requirements, making it possible to consider iterated rollouts[1]. With five iterations, a game, implemented in Java, takes about one hour and forty-five minutes on average on a SUN Blade 2000 machine with two 900MHz CPUs, and the probability of winning exceeds that of a human expert by about a factor of two. Our study represents an important contribution both to the study of the rollout method and to the study of solitaire.

## 2   Solitaire

It is one of the embarrassments of applied mathematics that we cannot determine the odds of winning the common game of solitaire. Many people play this game every day, yet simple questions such as *What is the chance of winning? How does this chance depend on the version I play? What is a good strategy?* remain beyond mathematical analysis.

According to Parlett [10], solitaire came into existence when fortune-telling with cards gained popularity in the eighteenth century. Many variations of solitaire exist today, such as Klondike, Freecell, and Carpet. Popularized by Microsoft Windows, Klondike has probably become the most widely played.

Klondike is played with a standard deck of cards: there are four suits (Spades, Clubs, Hearts, and Diamonds) each made up of thirteen cards ranked 1 through 13: Ace, 2, 3, ..., 10, Jack, Queen, and King. During the game, each card resides in one of thirteen *stacks*[2] : the *pile*, the *talon*, four *suit stacks* and seven *build stacks*. Each suit stack corresponds to a particular suit and build stacks are labeled 1 through 7.

At the beginning of the game, cards are dealt so that there is one card in the first build stack, two cards in the second build stack, ..., and seven cards in the seventh build stack. The top card on each of the seven build stacks is turned face-up while the rest of the cards in the build stacks face down. The other twenty-four cards, forming the pile, face down as well. The talon is initially empty.

The goal of the game is to move all cards into the suit stacks, aces first, then two's, and so on, with each suit stack evolving as an ordered increasing arrangement of cards of the same suit. The figure below shows a typical mid-game configuration.

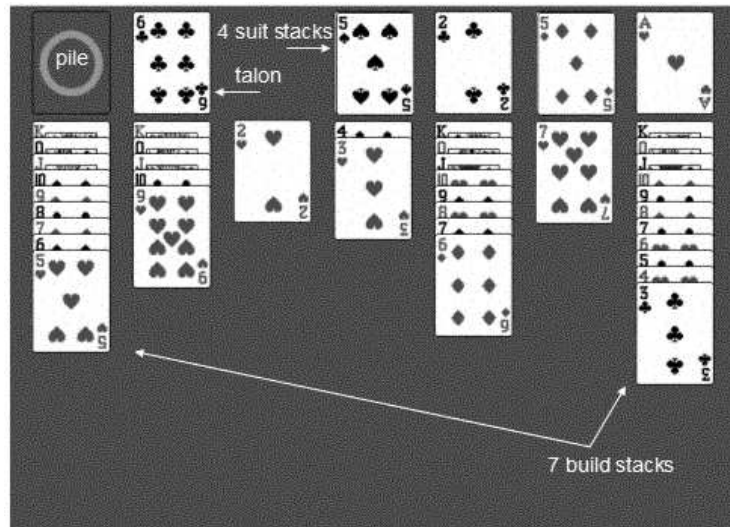

We will study a version of solitaire in which the identity of each card at each position is revealed to the player at the beginning of the game but the usual Klondike rules still apply. This version is played by a number of serious solitaire players as a much more difficult version than standard Klondike. Parlett [10] offers further discussion. We call this game *thoughtful solitaire* and now spell out the rules.

On each turn, the player can move cards from one stack to another in the following manner:

- Face-up cards of a build stack, called a *card block*, can be moved to the top of another build stack provided that the build stack to which the block is being moved accepts the block. Note that *all* face-up cards on the source stack must be moved together. After the move, these cards would then become the top cards of the stack to which they are moved, and their ordering is preserved. The card originally immediately beneath the card block, now the top card in its stack, is turned face-up. In the event that all cards in the source stack are moved, the player has an empty stack. [3]

- The top face-up card of a build stack can be moved to the top of a suit stack, provided that the suit stack accepts the card.

- The top card of a suit stack can be moved to the top of a build stack, provided that the build stack accepts the card.

- If the pile is not empty, a move can *deal* its top three cards to the talon, which maintains its cards in a first-in-last-out order. If the pile becomes empty, the player can *redeal* all the cards on the talon back to the pile in one card move. A redeal preserves the ordering of cards. The game allows an unlimited number of redeals.

- A card on the top of the talon can be moved to the top of a build stack or a suit stack, provided that the stack to which the card is being moved accepts the card.

- A build stack can only accept an incoming card block if the top card on the build stack is *adjacent* to and *braided* with the bottom card of the block. A card is *adjacent* to another card of rank $r$ if it is of rank $r + 1$. A card is *braided* with a card of suit $s$ if its suit is of a color different from $s$. Additionally, if a build stack is empty, it can only accept a card block whose bottom card is a King.
- A suit stack can only accept an incoming card of its corresponding suit. If a suit stack is empty, it can only accept an Ace. If it is not empty, the incoming card must be adjacent to the current top card of the suit stack.

As stated earlier, the objective is to end up with all cards on suit stacks. If this event occurs, the game is won.

## 3 Expert Play

We were introduced to thoughtful solitaire by a senior American mathematician (former president of the American Mathematical Society and indeed a famous combinatorialist) who had spent a number of years studying the game. He finds this version of solitaire much more thought-provoking and challenging than the standard Klondike. For instance, while the latter is usually played quickly, our esteemed expert averages about 20 minutes for each game of thoughtful solitaire. He carefully played and recorded 2,000 games, achieving a win rate of 36.6%.

With this background, it is natural to wonder how well an optimal player can perform at thoughtful solitaire. As we will illustrate, our best strategy offers a win rate of about 70%.

## 4 Machine Play

We have developed two strategies that play thoughtful solitaire. Both are based on the following general procedure:

1. Identify the set of legal moves.
2. Select and execute a legal move.
3. If all cards are on suit stacks, declare victory and terminate.
4. If the new card configuration repeats a previous one, declare loss and terminate [4].
5. Repeat procedure.

The only nontrivial task in this procedure is selection from the legal moves. We will first describe a heuristic strategy for selecting a legal move based on a card configuration. Afterwards, we will discuss the use of rollouts.

### 4.1 A Heuristic Strategy

Our heuristic strategy is based on part of the Microsoft Windows Klondike scoring system:

- The player starts the game with an initial score of 0.

- Whenever a card is moved from a build stack to a suit stack, the player gains 5 points.
- Whenever a card is moved from the talon to a build stack, the player gains 5 points.
- Whenever a card is moved from a suit stack to a build stack, the player loses 10 points.

In our heuristic strategy, we assign a score to each card move based on the above scoring system. We assign the score zero to any moves not covered by the above rules. When selecting a move, we choose among those that maximize the score.

Intuitively, this heuristic seems reasonable. The player has incentive to move cards from the talon to a build stack and from a build stack to a suit stack. One important element that the heuristic fails to capture, however, is what move to make when multiple moves maximize the score. Such decisions – especially during the early phases of a game – are crucial.

To select among moves that maximize score, we break the tie by assigning the following priorities:

- If the card move is from a build stack to another build stack, one of the following two assignments of priority occurs:
  - If the move turns an originally face-down card face-up, we assign this move a priority of $k + 1$, where $k$ is the number of originally face-down cards on the source stack before the move takes place.
  - If the move empties a stack, we assign this move a priority of 1.
- If the card move is from the talon to a build stack, one of the following three assignments of priority occurs:
  - If the card being moved is not a King, we assign the move priority 1.
  - If the card being moved is a King and its matching Queen is in the pile, in the talon, in a suit stack, or is face-up in a build stack, we assign the move priority 1.
  - If the card being moved is a King and its matching Queen is face-down in a build stack, we assign the move priority -1.
- For card moves not covered by the description above, we assign them a priority of 0.

In addition to introducing priorities, we modify the Windows Klondike scoring system further by adding the following change: in a card move, if the card being moved is a King and its matching Queen is face-down in a build stack, we assign the move a score of 0.

Note that given our assignment of scores and priorities, we practically disable card moves from a suit stack to a build stack. Because such moves have a negative score and a card move from the pile to the talon or from the talon to the pile has zero score and is almost always available, our strategy would always choose the pile-talon move over the moves from a suit stack to a build stack.

In the case when multiple moves equal in priority maximize the score, we randomly select a move among them.

The introduction of priority improves our original game-playing strategy in two ways: when we encounter a situation where we can move either one of two blocks on two separate build stacks atop the top card of a third build stack, we prefer moving the block whose stack has more face-down cards. Intuitively, such a move would strive to balance the number of face-down cards in stacks. Our experiments show that this heuristic significantly improves

success rate. The second way in which our prioritization scheme helps is that we are more deliberate in which King to select to enter an empty build stack. For instance, consider a situation where the King of Hearts and the King of Spades, both on the pile, are vying for an empty build stack and there is a face-up Queen of Diamonds on a build stack. We should certainly move the King of Spades to the empty build stack so that the Queen of Diamonds can be moved on top of it. Whereas our prioritization warrants such consideration, our original heuristic does not.

### 4.2 Rollouts

Consider a strategy $h$ that maps a card configuration $x$ to a legal move $h(x)$. What we described in the previous section was one example of a strategy $h$. In this section, we will discuss the rollout method as a procedure for amplifying the performance of any strategy. Given a strategy $h$, this procedure generates an improved strategy $h'$, called a *rollout strategy*. This idea was originally proposed by Tesauro and Galperin [13] and builds on the policy improvement algorithm of dynamic programming [1, 7].

Given a card configuration $x$. A strategy $h$ would make a move $h(x)$. A rollout strategy would make a move $h'(x)$, determined as follows:

1. For each legal move $a$, simulate the remainder of the game, taking move $a$ and then employing strategy $h$ thereafter.

2. If any of these simulations leads to victory, choose one of them randomly and let $h'(x)$ be the corresponding move $a$[5].

3. If none of the simulations lead to victory, let $h'(x) = h(x)$.

We can then iterate this procedure to generate a further improved strategy $h''$ that is a rollout strategy relative to $h'$. It is easy to prove that after a finite number of such iterations, we would arrive at an optimal strategy [2]. However, the computation time required grows exponentially in the number of iterations, so this may not be practical. Nevertheless, one might try a few iterations and hope that this offers the bulk of the mileage.

## 5   Results

We implemented in Java the heuristic strategy and the procedure for computing rollout strategies. Simulation results are provided in the following table and chart. We randomly generated a large number of games and played them with our algorithms in an effort to approximate the success probability with the percentage of games actually won. To determine a sufficient number of games to simulate, we used the Central Limit Theorem to compute the confidence bounds on success probability for each algorithm with a confidence level of 99%. For the original heuristic and 1 through 3 rollout iterations, we managed to achieve confidence bounds of [-1.4%, 1.4%]. For 4 and 5 rollout iterations, due to time constraints, we simulated fewer games and obtained weaker confidence bounds. Interestingly, however, after 5 rollout iterations, the resulting strategy wins almost twice as frequently as our esteemed mathematician.

| Player | Success Rate | Games Played | Average Time Per Game | 99% Confidence Bounds |
|---|---|---|---|---|
| Human expert | 36.6% | 2,000 | 20 minutes | ±2.78% |
| heuristic | 13.05% | 10,000 | .021 seconds | ±.882% |
| 1 rollout | 31.20% | 10,000 | .67 seconds | ±1.20% |
| 2 rollouts | 47.60% | 10,000 | 7.13 seconds | ±1.30% |
| 3 rollouts | 56.83% | 10,000 | 1 minute 36 seconds | ±1.30% |
| 4 rollouts | 60.51% | 1,000 | 18 minutes 7 seconds | ±4.00% |
| 5 rollouts | 70.20% | 200 | 1 hour 45 minutes | ±8.34% |

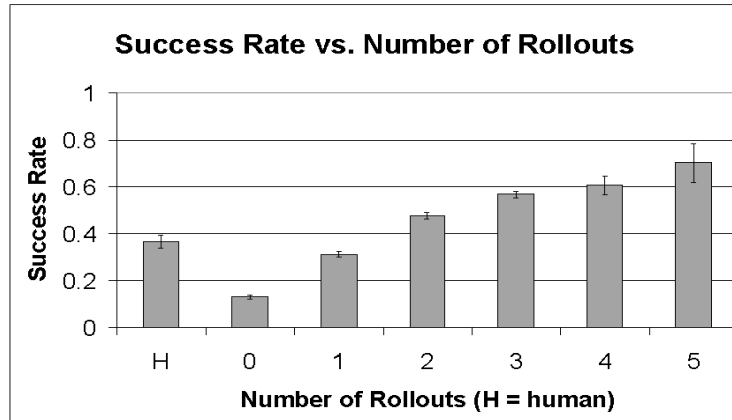

## 6 Future Challenges

One limitation of our rollout method lies in its recursive nature. Although it is clearly formulated and hence easily implemented in software, the algorithm does not provide a simple and explicit strategy for human players to make decisions.

One possible direction for further exploration would be to compute a value function, mapping the state of the game to an estimate of whether or not the game can be won. Certainly, this function could not be represented exactly, but we could try approximating it in terms of a linear combination of features of the game state, as is common in the approximate dynamic programming literature [2].

We have also attempted proving an upper bound for the success rate of thoughtful solitaire by enumerating sets of initial card configurations that would force loss. Currently, the tightest upper bound we can rigorously prove is 98.81%. Speed optimization of our software implementation is under way. If the success rate bound is improved and we are able to run additional rollout iterations, we may produce a verifiable near-optimal strategy for thoughtful solitaire.

## Acknowlegment

This material is based upon work supported by the National Science Foundation under Grant ECS-9985229.

## Footnotes

[1]Backgammon is stochastic because play is influenced by the roll of dice.

[2]In some solitaire literature, stacks are referred to as piles.

[3]It would seem to some that since the identity of all cards is revealed to the player, whether a card is face-up or face-down is irrelevant. We retain this property of cards as it is still important in describing the rules and formulating our strategy.

[4] One straight-forward way to determine if a card configuration has previously occurred is to store all encountered card configurations. Instead of doing so, however, we notice that there are three kinds of moves that could lead us into an infinite loop: pile-talon moves, moves that could juggle a card block between two build stacks, and moves that could juggle a card block between a build stack and a suit stack. Hence, to simplify our strategy, we disable the second kind of moves. Our heuristic will also practically disable the third kind. For the first kind, we record if any card move other than a pile-talon move has occurred since the last redeal. If not, we detect an infinite loop and declare loss.

[5]Note that at this stage, we could record all moves made in this simulation and declare victory. That is how our program is implemented. However, we leave step 2 as stated for the sake of clarity in presentation.

# References

[1] R. Bellman. *Applied Dynamic Programming.* Princeton University Press, 1957.

[2] D. Bertsekas and J.N. Tsitsiklis. *Neuro-Dynamic Programming.* Athena Scientific, 1996.

[3] D. P. Bertsekas, J. N. Tsitsiklis, and C. Wu, Rollout Algorithms for Combinatorial Optimization. *Journal of Heuristics*, 3:245-262, 1997.

[4] D. P. Bertsekas and D. A. Castañon. Rollout Algorithms for Stochastic Scheduling Problems. *Journal of Heuristics*, 5:89-108, 1999.

[5] D. Bertsimas and R. Demir. An Approximate Dynamic Programming Approach to Multi-dimensional Knapsack Problems. *Management Science*, 4:550-565, 2002.

[6] D. Bertsimas and I. Popescu. Revenue Management in a Dynamic Network Environment. *Transportation Science*, 37:257-277, 2003.

[7] R. Howard. *Dynamic Programming and Markov Processes.* MIT Press, 1960.

[8] A. McGovern, E. Moss, and A. Barto. Building a Basic Block Instruction Scheduler Using Reinforcement Learning and Rollouts. *Machine Learning*, 49:141-160, 2002.

[9] Y. Mansour and S. Singh. On the Complexity of Policy Iteration. In *Fifteenth Conference on Uncertainty in Artificial Intelligence*, 1999.

[10] D. Parlett. *A History of Card Games.* Oxford University Press, 1991.

[11] N. Secomandi. Analysis of a Rollout Approach to Sequencing Problems with Stochastic Routing Applications. *Journal of Heuristics*, 9:321-352, 2003.

[12] N. Secomandi. A Rollout Policy for the Vehicle Routing Problem with Stochastic Demands. *Operations Research*, 49:796-802, 2001.

[13] G. Tesauro and G. Galperin. On-line Policy Improvement Using Monte-Carlo Search. In *Advances in Neural Information Processing Systems*, 9:1068-1074, 1996.
